# A Parallel Gradient Descent Method for Learning in Analog VLSI Neural Networks

J. Alspector    R. Meir*    B. Yuhas    A. Jayakumar    D. Lippe†

Bellcore
Morristown, NJ 07962-1910

## Abstract

Typical methods for gradient descent in neural network learning involve calculation of derivatives based on a detailed knowledge of the network model. This requires extensive, time consuming calculations for each pattern presentation and high precision that makes it difficult to implement in VLSI. We present here a perturbation technique that *measures*, not *calculates*, the gradient. Since the technique uses the actual network as a measuring device, errors in modeling neuron activation and synaptic weights do not cause errors in gradient descent. The method is parallel in nature and easy to implement in VLSI. We describe the theory of such an algorithm, an analysis of its domain of applicability, some simulations using it and an outline of a hardware implementation.

## 1 Introduction

The most popular method for neural network learning is back-propagation (Rumelhart, 1986) and related algorithms that calculate gradients based on detailed knowledge of the neural network model. These methods involve calculating exact values of the derivative of the activation function. For analog VLSI implementations, such techniques require impossibly high precision in the synaptic weights and precise modeling of the activation functions. It is much more appealing to *measure* rather than *calculate* the gradient for analog VLSI implementation by perturbing either a

single weight (Jabri, 1991) or a single neuron (Widrow, 1990) and measuring the resulting change in the output error. However, perturbing only a single weight or neuron at a time loses one of the main advantages of implementing neural networks in analog VLSI, namely, that of computing weight changes in parallel. The one-weight-at-a-time perturbation method has the same order of time complexity as a serial computer simulation of learning. A mathematical analysis of the possibility of model free learning using parallel weight perturbations followed by local correlations suggests that random perturbations by additive, zero-mean, independent noise sources may provide a means of parallel learning (Dembo, 1990). We have previously used such a noise source (Alspector, 1991) in a different implementable learning model.

## 2   Gradient Estimation by Parallel Weight Perturbation

### 2.1   A Brownian Motion Algorithm

One can estimate the gradient of the error $E(\mathbf{w})$ with respect to any weight $w_1$ by perturbing $w_1$ by $\delta w_1$ and measuring the change in the output error $\delta E$ as the entire weight vector $\mathbf{w}$ except for component $w_1$ is held constant.

$$\frac{\delta E}{\delta w_1} = \frac{E(\mathbf{w} + \delta \mathbf{w_1}) - E(\mathbf{w})}{\delta w_1} \tag{1}$$

This leads to an approximation to the true gradient $\frac{\partial E}{\partial w_1}$:

$$\frac{\partial E}{\partial w_1} = \frac{\delta E}{\delta w_1} + O([\delta w_1]) \tag{2}$$

For small perturbations, the second (and higher order) term can be ignored. This method of perturbing weights one-at-a-time has the advantage of using the correct physical neurons and synapses in a VLSI implementation but has time complexity of $O(W)$ where $W$ is the number of weights.

Following (Dembo, 1990), let us now consider perturbing all weights simultaneously. However, we wish to have the perturbation vector $\delta \mathbf{w}$ chosen uniformly on a hypercube. Note that this requires only a random sign multiplying a fixed perturbation and is natural for VLSI. Dividing the resulting change in error by any single weight change, say $\delta w_1$, gives

$$\frac{\delta E}{\delta w_1} = \frac{E(\mathbf{w} + \delta \mathbf{w}) - E(\mathbf{w})}{\delta w_1} \tag{3}$$

which by a Taylor expansion is

$$\frac{\delta E}{\delta w_1} = \frac{\sum_{i=1}^{W} \frac{\partial E}{\partial w_i} \delta w_i}{\delta w_1} + O([\delta w_1]) \tag{4}$$

leading to the approximation (ignoring higher order terms)

$$\frac{\delta E}{\delta w_1} = \frac{\partial E}{\partial w_1} + \sum_{i>1}^{W} \left(\frac{\partial E}{\partial w_i}\right)\left(\frac{\delta w_i}{\delta w_1}\right). \tag{5}$$

An important point of this paper, emphasized by (Dembo, 1990) and embodied in Eq. (5), is that the last term has expectation value zero for random and independently distributed $\delta w_i$ since the last expression in parentheses is equally likely to be $+1$ as $-1$. Thus, one can approximately follow the gradient by perturbing all weights at the same time. If each synapse has access to information about the resulting change in error, it can adjust its weight by assuming it was the only weight perturbed. The weight change rule

$$\Delta w_i = -\eta \frac{\delta E}{\delta w_i}, \tag{6}$$

where $\eta$ is a learning rate, will follow the gradient on the average but with the considerable noise implied by the second term in Eq. (5). This type of stochastic gradient descent is similar to the random-direction Kiefer-Wolfowitz method (Kushner, 1978), which can be shown to converge under suitable conditions on $\eta$ and $\delta w_i$. This is also reminiscent of Brownian motion where, although particles may be subject to considerable random motion, there is a general drift of the ensemble of particles in the direction of even a weak external force. In this respect, there is some similarity to the directed drift algorithm of (Venkatesh, 1991), although that work applies to binary weights and single layer perceptrons whereas this algorithm should work for any level of weight quantization or precision - an important advantage for VLSI implementations - as well as any number of layers and even for recurrent networks.

## 2.2  Improving the Estimate by Multiple Perturbations

As was pointed out by (Dembo, 1990), for each pattern, one can reduce the variance of the noise term in Eq. (5) by repeating the random parallel perturbation many times to improve the statistical estimate. If we average over $P$ perturbations, we have

$$\frac{\delta E}{\delta w_1} = \frac{1}{P}\sum_{p=1}^{P}\frac{\delta E}{\delta w_1^p} = \frac{\partial E}{\partial w_1} + \frac{1}{P}\sum_{p=1}^{P}\sum_{i>1}^{W}\left(\frac{\partial E}{\partial w_i}\right)\left(\frac{\delta w_i^p}{\delta w_1^p}\right) \tag{7}$$

where $p$ indexes the perturbation number. The variance of the second term, which is a noise, $\nu$, is

$$<\nu^2> = \frac{1}{P^2}\sum_{p,p'=1}^{P}\sum_{i,i'>1}^{W}\left(\frac{\partial E}{\partial w_i}\right)\left(\frac{\partial E}{\partial w_{i'}}\right)\left\langle\left(\frac{\delta w_i^p}{\delta w_1^p}\right)\left(\frac{\delta w_{i'}^{p'}}{\delta w_1^{p'}}\right)\right\rangle \tag{8}$$

where the expectation value, $<>$, leads to the Kronecker delta function, $\delta_{ii'}^{pp'}$. This reduces Eq. (8) to

$$< \nu^2 > = \frac{1}{P^2} \sum_{p=1}^{P} \sum_{i>1}^{W} \left( \frac{\partial E}{\partial w_i} \right)^2. \tag{9}$$

The double sum over perturbations and weights (assuming the gradient is bounded and all gradient directions have the same order of magnitude) has magnitude $O(PW)$ so that the variance is $O(\frac{W}{P})$ and the standard deviation is

$$< (\nu^2) >^{\frac{1}{2}} = O\left( \left( \frac{W}{P} \right)^{\frac{1}{2}} \right). \tag{10}$$

Therefore, for a fixed variance in the noise term, it may be necessary to have a number of perturbations of the same order as the number of weights. So, if a high precision estimate of the gradient is needed throughout learning, it seems as though the time complexity will still be $O(W)$ giving no advantage over single perturbations. However, one or a few of the gradient derivatives may dominate the noise and reduce the effective number of parameters. One can also make a qualitative argument that early in learning, one does not need a precise estimate of the gradient since a general direction in weight space will suffice. Later, it will be necessary to make a more precise estimate for learning to converge.

### 2.3   The Gibbs Distribution and the Learning Problem

Note that the noise of Eq. (7) is gaussian since it is composed of a sum of random sign terms which leads to a binomial distribution and is gaussian distributed for large P. Thus, in the continuous time limit, the learning problem has Langevin dynamics such that the time rate of change of a weight $w_k$ is,

$$\frac{dw_k}{dt} = -\eta \frac{\delta E}{\delta w_k} = -\eta \frac{\partial E}{\partial w_k} + \nu_k, \tag{11}$$

and the learning problem converges in probability (Zinn-Justin, 1989), so that asymptotically $Pr(\mathbf{w}) \propto \exp[-\beta E(\mathbf{w})]$ where $\beta$ is inversely proportional to the noise variance.

Therefore, even though the gradient is noisy, one can still get a useful learning algorithm. Note that we can "anneal" $\nu_k$ by a variable perturbation method. Depending on the annealing schedule, this can result in a substantial speedup in learning over the one-weight-at-a-time perturbation technique.

### 2.4   Similar Work in these Proceedings

Coincidentally, there were three other papers with similar work at NIPS*92. This algorithm was presented with different approaches by both (Flower, 1993) and (Cauwenberghs, 1993). [1]   A continuous time version was implemented in VLSI but not on a neural network by (Kirk, 1993).

## 3    Simulations

### 3.1    Learning with Various Perturbation Iterations

We tried some simple problems using this technique in software. We used a standard sigmoid activation function with unit gain, a fixed size perturbation of .005 and random sign. The learning rate, $\eta$, was .1 and momentum, $\alpha$, was 0. We varied the number of perturbation iterations per pattern presentation from 1 to 128 ($2^l$ where $l$ varies from 0 to 7). We performed 10 runs for each condition and averaged the results. Fig. 1a shows the average learning curves for a 6 input, 12 hidden, 1 output unit parity problem as the number of perturbations per pattern presentation is varied. The symbol plotted is $l$.

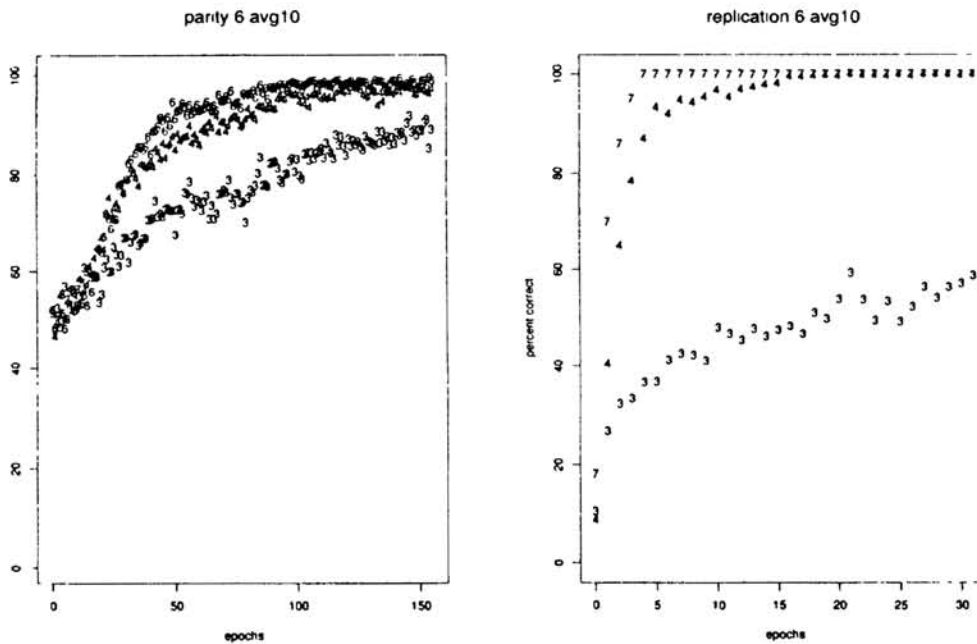

Figure 1. Learning curves for 6-12-1 parity and 6-6-6 replication.

There seems to be a critical number of perturbations, $P_c$, about 16 ($l = 4$) in this case, below which learning slows dramatically.

We repeated the measurements of Fig. 1a for different sizes of the parity problem using a N-2N-1 network. We also did these measurements on a different problem, replication or identity, where the task is to replicate the bit pattern of the input on the output. We used a N-N-N network for this task so that we have a comparison with the parity problem as N varies for roughly the same number of weights ($2N^2 + 2N$) in each network. The learning curves for the 6-6-6 problem are plotted in Fig. 1b. The critical value also seems to be 16 ($l = 4$).

---

perhaps because we do not decrease $\delta w$ and $\eta$ as learning proceeds. He did not check this for large problems as we did. In an implementation, one will not be able to reduce $\delta w$ too much so that the effect on the output error can be measured. It is also likely that multiple perturbations can be done more quickly than multiple pattern presentations, if learning speed is an issue. He also notes the importance of correlating with the change in error rather than the error alone as in (Dembo, 1990).

## 3.2  Scaling of the Critical Value with Problem Size

To determine how the critical value of perturbation iterations scales, we tried a variety of problems besides the N-N-N replication and N-2N-1 parity. We added N-2N-N replication and N-N-1 parity to see how more weights affect the same problem. We also did N-N-N/2 edge counting, where the output is the number of sign changes in an ordered row of N inputs. Finally we did N-2N-N and N-N-N hamming where the output is the closest hamming code for N inputs. We varied the number of perturbation iterations so that p = 1,2,5,10,20,50,100,200,400.

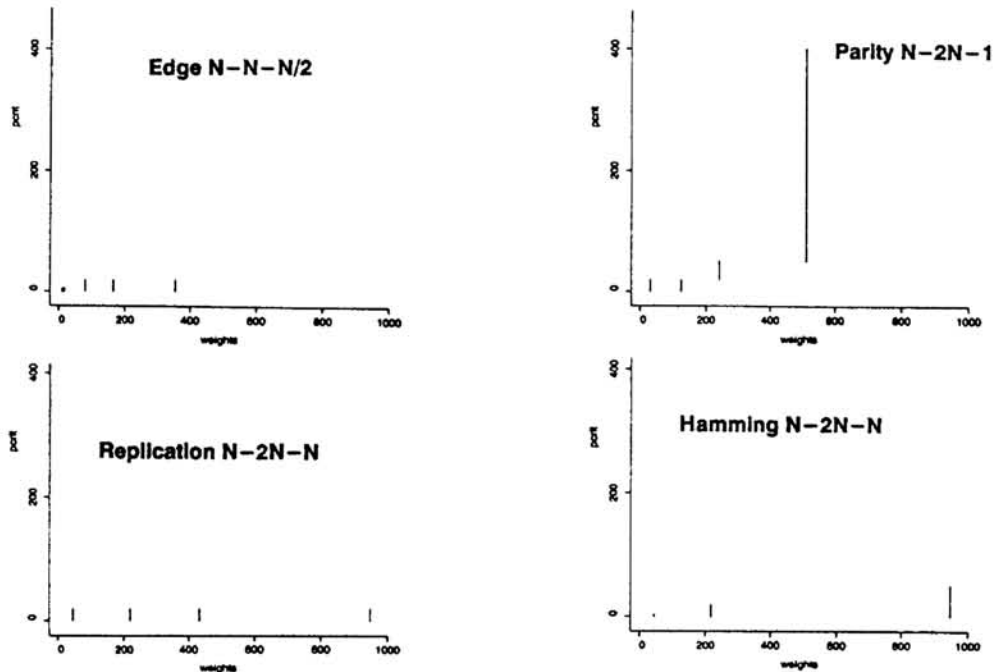

Figure 2. Critical value scaling for different problems.

Fig. 2 gives a feel for the effective scale of the problem by plotting the critical value of the number of perturbation iterations as a function of the number of weights for some of the problems we looked at. Note that the required number of iterations is not a steep function of the network size except for the parity problem. We speculate that the scaling properties are dependent on the shape of the error surface. If the derivatives in Eq. 9 are large in all dimensions (learning on a bowl-shaped surface), then the effective number of parameters is large and the variance of the noise term will be on the order of the number of weights, leading to a steep dependence in Fig. 2. If, however, there are only a few weight directions with significantly large error derivatives (learning on a taco shell), then the noise will scale at a slower rate than the number of weights leading to a weak dependence of the critical value with problem size. This is actually a nice feature of parallel perturbative learning because it means learning will be noisy and slow in a bowl where it's easy, but precise and fast in a taco shell where it's hard.

The critical value is required for convergence at the end of learning but not at the start. This means it should be possible to anneal the number of perturbation iterations to achieve an additional speedup over the one-weight-at-a-time perturba-

tion technique. We would also like to understand how to vary $\delta w$ and $\eta$ as learning proceeds. The stochastic approximation literature is likely to serve as a useful guide.

## 3.3    Computational Geometry of Stochastic Gradient Descent

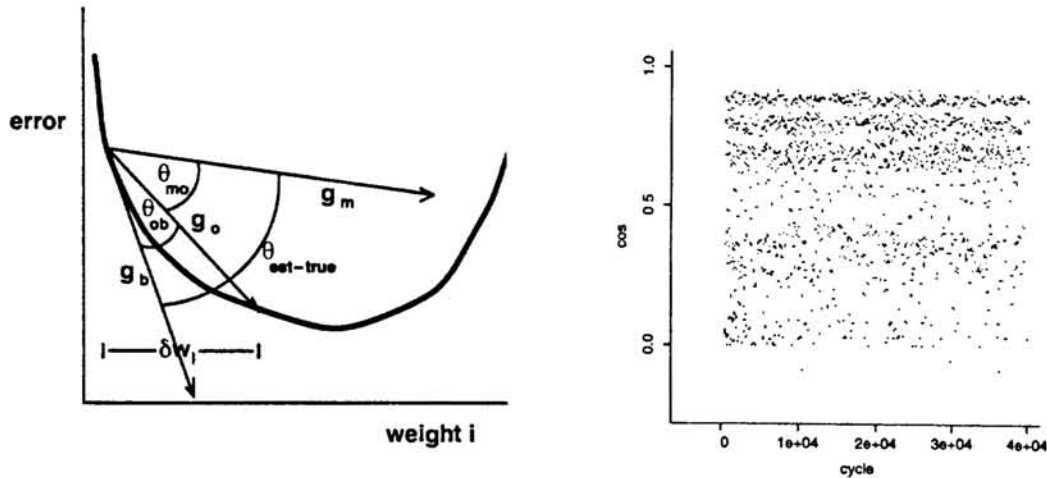

Figure 3. Computational Geometry of Stochastic Gradient Descent.

Fig. 3a shows some relevant gradient vectors and angles in the learning problem. For a particular pattern presentation, the true gradient, $g_b$, from a back-propagation calculation is compared with the one-weight-at-a-time gradient, $g_o$, from a perturbation, $\delta w_i$, in one weight direction. The gradient from perturbing all weights, $g_m$, adds a noise vector to $g_o$. By taking the normalized dot product between $g_m$ and $g_b$, one obtains the direction cosine between the estimated and the true gradient direction. This is plotted in Fig. 3b for the 10 input N-N-1 parity problem for all nine perturbation values. The shaded bands increase in $cos$ (decrease in angle) as the number of perturbations goes from 1 to 400. Note that the angles are large but that learning still takes place. Note also that the dot product is almost always positive except for a few points at low perturbation numbers. Incidentally, by looking at plots of the true to one-weight-at-a-time angles (not shown), we see that the large angles are due almost entirely to the parallel perturbative noise term and not to the stepsize, $\delta w$.

## 4    Outline of an analog implementation

Fig. 4 shows a diagram of a learning synapse using this perturbation technique. Note that its only inputs are a single bit representing the sign of the perturbation and a broadcast signal representing the change in the output error. Multiple perturbations can be averaged by the summing buffer and weight is stored as charge on a capacitor or floating gate device.

An estimate of the power and area of an analog chip implementation gives the following: Using a standard 1.2μm, double poly technology, the synapse with about 7 to 8 bits of resolution and which includes a 0.5 pf storage capacitor, weight refresh (Hochet, 1989) and update circuitry can be fabricated with an area of about 1600 $\mu m^2$ and with a power dissipation of about 100 $\mu$W with continuous self-refresh. This translates into a chip of about 22000 synapses at 2.2 watts on a 36 $mm^2$ die core. It is likely that the power requirements can be greatly reduced with a more relaxed refresh technique or with a suitable non-volatile analog storage technology.

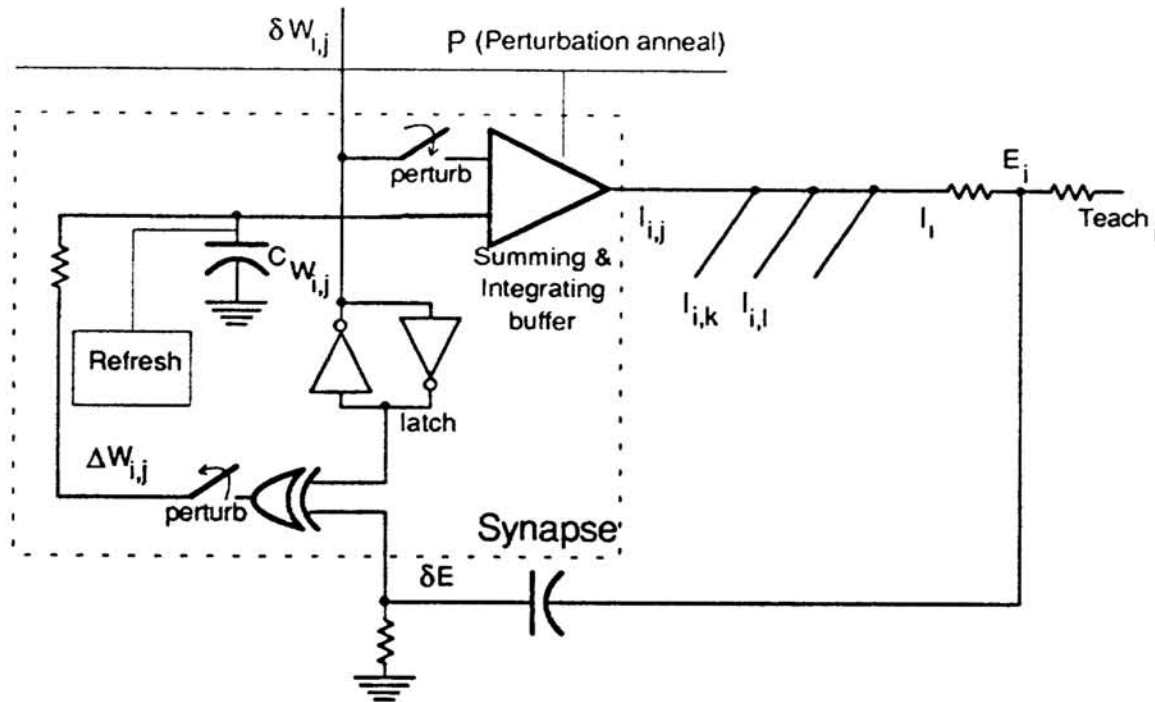

Figure 4. Diagram of perturbative learning synapse.

We intend to use our noise generation technique (Alspector, 1991) to provide uncorrelated perturbations potentially to thousands of synapses. Note also that the error signal can be generated by a simple resistor or a comparator followed by a summer. The difference signal can be generated by a simple differentiator.

## 5  Conclusion

We have analyzed a parallel perturbative learning technique and shown that it should converge under the proper conditions. We have performed simulations on a variety of test problems to demonstrate the scaling behavior of this learning algorithm. We are continuing work to understand speedups possible in an analog VLSI implementation. Finally, we describe such an implementation. Future work will involve applying this technique to learning in recurrent networks.

**Acknowledgment**

We thank Barak Pearlmutter for valuable and insightful discussions and Gert Cauwenberghs for making an advance copy of his paper available. This work has

been partially supported by AFOSR contract F49620-90-C-0042, DEF.

## Footnotes

*Present address: Dept. of EE; Technion; Haifa, Israel

†Present address: Dept. of EE; MIT; Cambridge, MA

[1] We note that (Cauwenberghs, 1993) shows that multiple perturbations are not needed for learning if $\Delta w$ is small enough and he does not study them. This does not agree with our simulations (following)

## References

J. Alspector, J. W. Gannett, S. Haber, M.B. Parker, and R. Chu, "A VLSI-Efficient Technique for Generating Multiple Uncorrelated Noise Sources and Its Application to Stochastic Neural Networks", *IEEE Trans. Circuits and Systems*, **38**, 109, (Jan., 1991).

J. Alspector, A. Jayakumar, and S. Luna, "Experimental Evaluation of Learning in a Neural Microsystem" in *Advances in Neural Information Processing Systems 4*, J. E. Moody, S. J. Hanson, and R. P. Lippmann (eds.) San Mateo,CA: Morgan-Kaufmann Publishers (1992), pp. 871-878.

G. Cauwenberghs, "A Fast Stochastic Error-Descent Algorithm for Supervised Learning and Optimization," in *Advances in Neural Information Processing Systems,* San Mateo, CA: Morgan Kaufman Publishers, vol. **5**, 1993.

A. Dembo and T. Kailath, "Model-Free Distributed Learning", *IEEE Trans. Neural Networks* **B1**, (1990) pp. 58-70.

B. Flower and M. Jabri, "Summed Weight Neuron Perturbation: An $\mathcal{O}(n)$ Improvement over Weight Perturbation," in *Advances in Neural Information Processing Systems,* San Mateo, CA: Morgan Kaufman Publishers, vol. **5**, 1993.

B. Hochet, "Multivalued MOS memory for Variable Synapse Neural Network", *Electronics Letters*, vol 25, no 10, (May 11, 1989) pp. 669-670.

M. Jabri and B. Flower, "Weight Perturbation: An Optimal Architecture and Learning Technique for Analog VLSI Feedforward and Recurrent Multilayer Networks", *Neural Computation* **3** (1991) pp. 546-565.

D. Kirk, D. Kerns, K. Fleischer, and A. Barr, "Analog VLSI Implementation of Gradient Descent," in *Advances in Neural Information Processing Systems,* San Mateo, CA: Morgan Kaufman Publishers, vol. **5**, 1993.

H.J. Kushner and D.S. Clark, "Stochastic Approximation Methods for Constrained and Unconstrained Systems", p. 58 ff., Springer-Verlag, New York, (1978).

D. E. Rumelhart, G. E. Hinton, and R. J. Williams, "Learning Internal Representations by Error Propagation", in *Parallel Distributed Processing: Explorations in the Microstructure of Cognition. Vol. 1: Foundations*, D. E. Rumelhart and J. L. McClelland (eds.), MIT Press, Cambridge, MA (1986), p. 318.

S. Venkatesh, "Directed Drift: A New Linear Threshold Algorithm for Learning Binary Weights On-Line", *Journal of Computer Science and Systems*, (1993), in press.

B. Widrow and M. A. Lehr, "30 years of Adaptive Neural Networks. Perceptron, Madaline, and Backpropagation", *Proc. IEEE* **78** (1990) pp. 1415-1442.

J. Zinn-Justin, "Quantum Field Theory and Critical Phenomena", p. 57 ff., Oxford University Press, New York, (1989).
